# Independent Components Analysis through Product Density Estimation

**Trevor Hastie and Rob Tibshirani**
Department of Statistics
Stanford University
Stanford, CA, 94305
{*hastie,tibs*}*@stat.stanford.edu*

## Abstract

We present a simple direct approach for solving the ICA problem, using density estimation and maximum likelihood. Given a candidate orthogonal frame, we model each of the coordinates using a semi-parametric density estimate based on cubic splines. Since our estimates have two continuous derivatives, we can easily run a second order search for the frame parameters. Our method performs very favorably when compared to state-of-the-art techniques.

## 1 Introduction

Independent component analysis (ICA) is a popular enhancement over principal component analysis (PCA) and factor analysis. In its simplest form, we observe a random vector $X \in \mathbb{R}^p$ which is assumed to arise from a linear mixing of a latent random source vector $S \in \mathbb{R}^p$,

$$(1) \qquad\qquad X = \mathbf{A}S;$$

the components $S_j$, $j = 1, \ldots, p$ of $S$ are assumed to be independently distributed. The classical example of such a system is known as the "cocktail party" problem. Several people are speaking, music is playing, etc., and microphones around the room record a mix of the sounds. The ICA model is used to extract the original sources from these different mixtures.

Without loss of generality, we assume $\mathrm{E}(S) = 0$ and $\mathrm{Cov}(S) = \mathbf{I}$, and hence $\mathrm{Cov}(X) = \mathbf{A}\mathbf{A}^T$. Suppose $S^* = \mathbf{R}S$ represents a transformed version of $S$, where $\mathbf{R}$ is $p \times p$ and orthogonal. Then with $\mathbf{A}^* = \mathbf{A}\mathbf{R}^T$ we have $X^* = \mathbf{A}^*S^* = \mathbf{A}\mathbf{R}^T\mathbf{R}S = X$. Hence the second order moments $\mathrm{Cov}(X) = \mathbf{A}\mathbf{A}^T = \mathbf{A}^*\mathbf{A}^{*T}$ do not contain enough information to distinguish these two situations.

Model (1) is similar to the factor analysis model (Mardia, Kent & Bibby 1979), where $S$ and hence $X$ are assumed to have a Gaussian density, and inference is typically based on the likelihood of the observed data. The factor analysis model typically has fewer than $p$ components, and includes an error component for each variable. While similar modifications are possible here as well, we focus on the full-component model in this paper. Two facts are clear:

- Since a multivariate Gaussian distribution is completely determined by its first and second moments, this model would not be able to distinguish $\mathbf{A}$ and $\mathbf{A}^*$. Indeed, in factor analysis one chooses from a family of factor rotations to select a suitably interpretable version.

- Multivariate Gaussian distributions are completely specified by their second-order moments. If we hope to recover the original $\mathbf{A}$, at least $p-1$ of the components of $S$ will have to be non-Gaussian.

Because of the lack of information in the second moments, the first step in an ICA model is typically to transform $X$ to have a scalar covariance, or to *pre-whiten* the data. From now on we assume $\mathrm{Cov}(X) = \mathbf{I}$, which implies that $\mathbf{A}$ is orthogonal.

Suppose the density of $S_j$ is $f_j$, $j = 1,\ldots,p$, where at most one of the $f_j$ are Gaussian. Then the joint density of $S$ is

$$(2) \qquad f_S(s) = \prod_{j=1}^{p} f_j(s_j),$$

and since $\mathbf{A}$ is orthogonal, the joint density of $X$ is

$$(3) \qquad f_X(x) = \prod_{j=1}^{p} f_j(a_j^T x),$$

where $a_j$ is the $j$th column of $\mathbf{A}$. Equation (3) follows from $S = \mathbf{A}^T X$ due to the orthogonality of $\mathbf{A}$, and the fact that the determinant in this multivariate transformation is 1.

In this paper we fit the model (3) directly using semi-parametric maximum likelihood. We represent each of the densities $f_j$ by an *exponentially tilted Gaussian* density (Efron & Tibshirani 1996).

$$(4) \qquad f_j(s_j) = \phi(s_j) e^{g_j(s_j)},$$

where $\phi$ is the standard univariate Gaussian density, and $g_j$ is a smooth function, restricted so that $f_j$ integrates to 1. We represent each of the functions $g_j$ by a cubic smoothing spline, a rich class of smooth functions whose roughness is controlled by a penalty functional. These choices lead to an attractive and effective semi-parametric implementation of ICA:

- Given $\mathbf{A}$, each of the components $f_j$ in (3) can be estimated separately by maximum likelihood. Simple algorithms and standard software are available.

- The components $g_j$ represent departures from Gaussianity, and the expected log-likelihood ratio between model (3) and the gaussian density is given by $E_X \sum_j g_j(a_j^T X)$, a flexible *contrast* function.

- Since the first and second derivatives of each of the estimated $g_j$ are immediately available, second order methods are available for estimating the orthogonal matrix $\mathbf{A}$. We use the fixed point algorithms described in (Hyvärinen & Oja 1999).

- Our representation of the $g_j$ as smoothing splines casts the estimation problem as density estimation in a reproducing kernel Hilbert space, an infinite family of smooth functions. This makes it directly comparable with the "Kernel ICA" approach of Bach & Jordan (2001), with the advantage that we have $O(N)$ algorithms available for the computation of our contrast function, and its first two derivatives.

In the remainder of this article, we describe the model in more detail, and evaluate its performance on some simulated data.

## 2 Fitting the Product Density ICA model

Given a sample $x_1, \ldots, x_N$ we fit the model (3),(4) by maximum penalized likelihood. The data are first transformed to have zero mean vector, and identity covariance matrix using the singular value decomposition. We then maximize the criterion

$$(5) \qquad \sum_{j=1}^{p} \left\{ \frac{1}{N} \sum_{i=1}^{N} \left[ \log \phi(a_j^T x_i) + g_j(a_j^T x_i) \right] - \lambda_j \int g_j''^2(t) dt \right\}$$

subject to

$$(6) \qquad\qquad a_j^T a_k \;=\; \delta_{jk} \; \forall j, k$$

$$(7) \qquad\qquad \int \phi(s) e^{g_j(s)} ds \;=\; 1 \; \forall j$$

For fixed $a_j$ and hence $s_{ij} = a_j^T x_i$ the solutions for $g_j$ are known to be cubic splines with knots at each of the unique values of $s_{ij}$ (Silverman 1986). The $p$ terms decouple for fixed $a_j$, leaving us $p$ separate penalized density estimation problems. We fit the functions $g_j$ and directions $a_j$ by optimizing (5) in an alternating fashion, as described in Algorithm 1. In step (a), we find the optimal $g_j$ for fixed $g_j$; in

---

**Algorithm 1** Product Density ICA algorithm

1. Initialize **A** (random Gaussian matrix followed by orthogonalization).
2. Alternate until convergence of **A**, using the Amari metric (16).
   - (a) Given **A**, optimize (5) w.r.t. $g_j$ (separately for each $j$), using the penalized density estimation algorithm 2.
   - (b) Given $g_j$, $j = 1, \ldots, p$, perform one step of the fixed point algorithm 3 towards finding the optimal **A**.

---

step (b), we take a single fixed-point step towards the optimal **A**. In this sense Algorithm 1 can be seen to be maximizing the *profile* penalized log-likelihood w.r.t. **A**.

### 2.1 Penalized density estimation

We focus on a single coordinate, with $N$ observations $s_i$, $i = 1, \ldots, N$ (where $s_i = a_k^T x_i$ for some $k$). We wish to maximize

$$(8) \qquad \frac{1}{N} \sum_{i=1}^{N} \{ \log \phi(s_i) + g(s_i) \} - \lambda \int g''^2(s) ds$$

subject to $\int \phi(s) e^{g(s)} ds = 1$. Silverman (1982) shows that one can incorporate the integration constraint by using the modified criterion (without a Lagrange multiplier)

$$(9) \qquad \frac{1}{N} \sum_{i=1}^{N} \{ \log \phi(s_i) + g(s_i) \} - \int \phi(s) e^{g(s)} ds - \lambda \int g''^2(s) ds.$$

Since (9) involves an integral, we need an approximation. We construct a fine grid of $L$ values $s_\ell^*$ in increments $\Delta$ covering the observed values $s_i$, and let

$$(10) \qquad y_\ell^* = \frac{\#s_i \in (s_\ell^* - \Delta/2, s_\ell^* + \Delta/2)}{N}$$

Typically we pick $L$ to be 1000, which is more than adequate. We can then approximate (9) by

$$(11) \qquad \sum_{\ell=1}^{L} \left\{ y_i^* \left[ \log(\phi(s_\ell^*)) + g(s_\ell^*) \right] - \Delta\phi(s_\ell^*)e^{g(s_\ell^*)} \right\} - \lambda \int g''^2(s)ds.$$

This last expression can be seen to be proportional to a penalized Poisson log-likelihood with response $y_\ell^*/\Delta$ and penalty parameter $\lambda/\Delta$, and mean $\mu(s) = \phi(s)e^{g(s)}$. This is a *generalized additive model* (Hastie & Tibshirani 1990), with an *offset* term $\log(\phi(s))$, and can be fit using a Newton algorithm in $O(L)$ operations. As with other GAMs, the Newton algorithm is conveniently re-expressed as an iteratively reweighted penalized least squares regression problem, which we give in Algorithm 2.

---

**Algorithm 2** Iteratively reweighted penalized least squares algorithm for fitting the tilted Gaussian spline density model.

1. Initialize $g \equiv 0$.

2. Repeat until convergence:

   (a) Let $\mu(s_\ell^*) = \phi(s_\ell^*)e^{g(s_\ell^*)}$, $\ell = 1, \ldots, L$, and $w_\ell = \mu(s_\ell^*)$.

   (b) Define the working response

$$(12) \qquad z_\ell = g(s_\ell^*) + \frac{y_\ell^* - \mu(s_\ell^*)}{\mu(s_\ell^*)}$$

   (c) Update g by solving the weighted penalized least squares problem

$$(13) \qquad \min_g \sum_{\ell=1}^{L} w_\ell(z_\ell - g(s_\ell^*))^2 + \frac{2\lambda}{\Delta} \int g''(s)^2 ds.$$

   This amounts to fitting a weighted smoothing spline to the pairs $(s_\ell^*, z_\ell)$ with weights $w_\ell$ and tuning parameter $2\lambda/\Delta$.

---

Although other semi-parametric regression procedures could be used in (13), the cubic smoothing spline has several advantages:

- It has knots at all $L$ of the pseudo observation sites $s_\ell^*$. The values $s_\ell^*$ can be fixed for all terms in the model (5), and so a certain amount of pre-computation can be performed. Despite the large number of knots and hence basis functions, the local support of the B-spline basis functions allows the solution to (13) to be obtained in $O(L)$ computations.

- The first and second derivatives of $g$ are immediately available, and are used in the second-order search for the direction $a_j$ in Algorithm 1.

- As an alternative to choosing a value for $\lambda$, we can control the amount of smoothing through the *effective number of parameters*, given by the trace of the linear operator matrix implicit in (13) (Hastie & Tibshirani 1990).

- It can also be shown that because of the form of (9), the resulting density inherits the mean and variance of the data (0 and 1); details will be given in a longer version of this paper.

## 2.2  A fixed point method for finding the orthogonal frame

For fixed functions $g_j$, the penalty term in (5) does not play a role in the search for $\mathbf{A}$. Since all of the columns $a_j$ of any $\mathbf{A}$ under consideration are mutually orthogonal and unit norm, the Gaussian component

$$\sum_{j=1}^{p} \log \phi(a_j^T x_i) = (2\pi)^{-p/2} e^{-x_i^T \mathbf{A}\mathbf{A}^T x_i / 2}$$

$$= (2\pi)^{-p/2} e^{-x_i^T x_i / 2}$$

does not depend on $\mathbf{A}$. Hence what remains to be optimized can be seen as the log-likelihood ratio between the fitted model and the Gaussian model, which is simply

$$(14) \qquad C(\mathbf{A}) = \frac{1}{N} \sum_{i=1}^{N} \sum_{j=1}^{p} g_j(a_j^T x_i)$$

$$= \sum_{j=1}^{p} C_j(a_j)$$

Since the choice of each $g_j$ improves the log-likelihood relative to the Gaussian, it is easy to show that $C(\mathbf{A})$ is positive and zero only if, for the particular value of $\mathbf{A}$, the log-likelihood cannot distinguish the tilted model from a Gaussian model. $C(\mathbf{A})$ has the form of a sum of contrast functions for detecting departures from Gaussianity. Hyvärinen, Karhunen & Oja (2001) refer to the expected log-likelihood ratio as the *negentropy*, and use simple contrast functions to approximate it in their *FastICA* algorithm. Our regularized approach can be seen as a way to construct a flexible contrast function adaptively using a large set of basis functions.

---

**Algorithm 3** Fixed point update for **A**.

1. For $j = 1, \ldots, p$:

   $$(15) \qquad a_j \leftarrow \mathrm{E}\left\{ X g_j'(a_j^T X) - \mathrm{E}\{g_j''(a_j^T X)\} a_j \right\},$$

   where E represents expectation w.r.t. the sample $x_i$, and $a_j$ is the $j$th column of **A**.

2. Orthogonalize **A**: Compute its SVD, $\mathbf{A} = \mathbf{U}\mathbf{D}\mathbf{V}^T$, and replace $\mathbf{A} \leftarrow \mathbf{U}\mathbf{V}^T$.

---

Since we have first and second derivatives avaiable for each $g_j$, we can mimic exactly the fast fixed point algorithm developed in (Hyvärinen et al. 2001, page 189); see algorithm 3. Figure 1 shows the optimization criterion $C$ (14) above, as well as the two criteria used to approximate negentropy in FastICA by Hyvärinen et al. (2001) [page 184]. While the latter two agree with $C$ quite well for the uniform example (left panel), they both fail on the mixture-of-Gaussians example, while $C$ is also successful there.

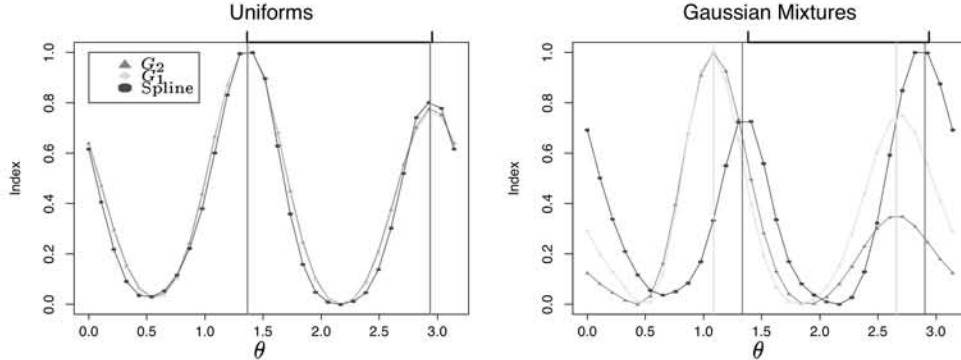

Figure 1: The optimization criteria and solutions found for two different examples in $\mathbb{R}^2$ using *FastICA* and our *ProDenICA*. $G_1$ and $G_2$ refer to the two functions used to define *negentropy* in FastICA. In the left example the independent components are uniformly distributed, in the right a mixture of Gaussians. In the left plot, all the procedures found the correct frame; in the right plot, only the spline based approach was successful. The vertical lines indicate the solutions found, and the two tick marks at the top of each plot indicate the true angles.

# 3   Comparisons with fast ICA

In this section we evaluate the performance of the product density approach (*ProDenICA*), by mimicking some of the simulations performed by Bach & Jordan (2001) to demonstrate their *Kernel ICA* approach. Here we compare *ProDenICA* only with *FastICA*; a future expanded version of this paper will include comparisons with other ICA procedures as well.

The left panel in Figure 2 shows the 18 distributions used as a basis of comparison. These exactly or very closely approximate those used by Bach & Jordan (2001). For each distribution, we generated a pair of independent components ($N$=1024), and a random mixing matrix in $\mathbb{R}^2$ with condition number between 1 and 2. We used our Splus implementation of the *FastICA* algorithm, using the negentropy criterion based on the nonlinearity $G_1(s) = \log\cosh(s)$, and the symmetric orthogonalization scheme as in Algorithm 3 (Hyvärinen et al. 2001, Section 8.4.3). Our *ProDenICA* method is also implemented in Splus. For both methods we used five random starts (without iterations). Each of the algorithms delivers an orthogonal mixing matrix $\mathbf{A}$ (the data were *pre-whitened*), which is available for comparison with the generating orthogonalized mixing matrix $\mathbf{A}_0$. We used the Amari metric(Bach & Jordan 2001) as a measure of the closeness of the two frames:

$$(16) \qquad d(\mathbf{A}_0, \mathbf{A}) = \frac{1}{2p} \sum_{i=1}^{p} \left( \frac{\sum_{j=1}^{p} |r_{ij}|}{\max_j |r_{ij}|} - 1 \right) + \frac{1}{2p} \sum_{j=1}^{p} \left( \frac{\sum_{i=1}^{p} |r_{ij}|}{\max_i |r_{ij}|} - 1 \right),$$

where $r_{ij} = (\mathbf{A}_o \mathbf{A}^{-1})_{ij}$. The right panel in Figure 2 shows boxplots of the pairwise differences $d(\mathbf{A}_0, \mathbf{A}_F) - d(\mathbf{A}_0, \mathbf{A}_P)$ ($\times 100$), where the subscripts denote *ProDenICA* or *FastICA*. *ProDenICA* is competitive with *FastICA* in all situations, and dominates in most of the mixture simulations. The average Amari error ($\times 100$) for *FastICA* was 13.4 (2.7), compared with 3.0 (0.4) for *ProDenICA* (Bach & Jordan (2001) report averages of 6.2 for *FastICA*, and 3.8 and 2.9 for their two *KernelICA* methods).

We also ran 300 simulations in $\mathbb{R}^4$, using $N = 1000$, and selecting four of the

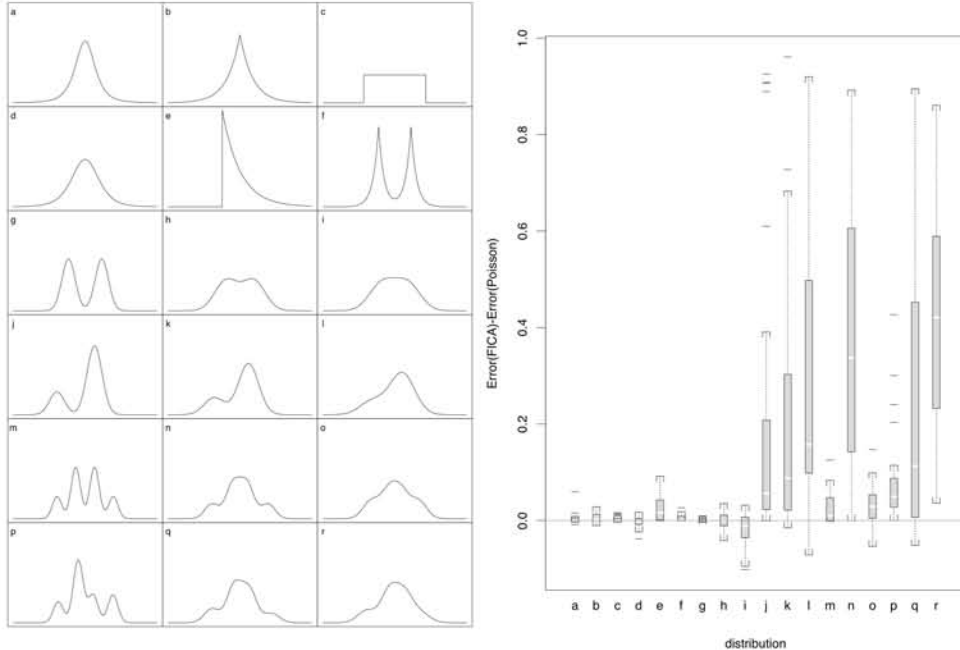

Figure 2: The left panel shows eighteen distributions used for comparisons. These include the "t", uniform, exponential, mixtures of exponentials, symmetric and asymmetric gaussian mixtures. The right panel shows boxplots of the improvement of *ProDenICA* over *FastICA* in each case, using the Amari metric, based on 30 simulations in $\mathbb{R}^2$ for each distribution.

18 distributions at random. The average Amari error ($\times 100$) for *FastICA* was 26.1 (1.5), compared with 9.3 (0.6) for *ProDenICA* (Bach & Jordan (2001) report averages of 19 for *FastICA*, and 13 and 9 for their two *KernelICA* methods).

## 4    Discussion

The ICA model stipulates that after a suitable orthogonal transformation, the data are independently distributed. We implement this specification directly using semi-parametric product-density estimation. Our model delivers estimates of both the mixing matrix **A**, and estimates of the densities of the independent components.

Many approaches to ICA, including *FastICA*, are based on minimizing approximations to entropy. The argument, given in detail in Hyvärinen et al. (2001) and reproduced in Hastie, Tibshirani & Friedman (2001), starts with minimizing the mutual information — the KL divergence between the full density and its independence version. *FastICA* uses very simple approximations based on single (or a small number of) non-linear contrast functions, which work well for a variety of situations, but not at all well for the more complex gaussian mixtures. The log-likelihood for the spline-based product-density model can be seen as a direct estimate of the mutual information; it uses the empirical distribution of the observed data to represent their joint density, and the product-density model to represent the independence density. This approach works well in both the simple and complex situations automatically, at a very modest increase in computational effort. As a side benefit,

the form of our tilted Gaussian density estimate allows our log-likelihood criterion to be interpreted as an estimate of negentropy, a measure of departure from the Gaussian.

Bach & Jordan (2001) combine a nonparametric density approach (via reproducing kernel Hilbert function spaces) with a complex measure of independence based on the maximal correlation. Their procure requires $O(N^3)$ computations, compared to our $O(N)$. They motivate their independence measures as approximations to the mutual independence. Since the smoothing splines are exactly function estimates in a RKHS, our method shares this flexibility with their Kernel approach (and is in fact a "Kernel" method). Our objective function, however, is a much simpler estimate of the mutual information. In the simulations we have performed so far, it seems we achieve comparable accuracy.

# References

Bach, F. & Jordan, M. (2001), Kernel independent component analysis, Technical Report UCB/CSD-01-1166, Computer Science Division, University of California, Berkeley.

Efron, B. & Tibshirani, R. (1996), 'Using specially designed exponential families for density estimation', *Annals of Statistics* **24**(6), 2431–2461.

Hastie, T. & Tibshirani, R. (1990), *Generalized Additive Models*, Chapman and Hall.

Hastie, T., Tibshirani, R. & Friedman, J. (2001), *The Elements of Statistical Learning; Data mining, Inference and Prediction*, Springer Verlag, New York.

Hyvärinen, A., Karhunen, J. & Oja, E. (2001), *Independent Component Analysis*, Wiley, New York.

Hyvärinen, A. & Oja, E. (1999), 'Independent component analysis: Algorithms and applications', *Neural Networks* .

Mardia, K., Kent, J. & Bibby, J. (1979), *Multivariate Analysis*, Academic Press.

Silverman, B. (1982), 'On the estimation of a probability density function by the maximum penalized likelihood method', *Annals of Statistics* **10**(3), 795–810.

Silverman, B. (1986), *Density Estimation for Statistics and Data Analysis*, Chapman and Hall.
